# A Solution for Missing Data in Recurrent Neural Networks With an Application to Blood Glucose Prediction

**Volker Tresp and Thomas Briegel** *
Siemens AG
Corporate Technology
Otto-Hahn-Ring 6
81730 München, Germany

## Abstract

We consider neural network models for stochastic nonlinear dynamical systems where measurements of the variable of interest are only available at irregular intervals i.e. most realizations are missing. Difficulties arise since the solutions for prediction and maximum likelihood learning with missing data lead to complex integrals, which even for simple cases cannot be solved analytically. In this paper we propose a specific combination of a nonlinear recurrent neural predictive model and a linear error model which leads to tractable prediction and maximum likelihood adaptation rules. In particular, the recurrent neural network can be trained using the real-time recurrent learning rule and the linear error model can be trained by an EM adaptation rule, implemented using forward-backward Kalman filter equations. The model is applied to predict the glucose/insulin metabolism of a diabetic patient where blood glucose measurements are only available a few times a day at irregular intervals. The new model shows considerable improvement with respect to both recurrent neural networks trained with teacher forcing or in a free running mode and various linear models.

## 1 INTRODUCTION

In many physiological dynamical systems measurements are acquired at irregular intervals. Consider the case of blood glucose measurements of a diabetic who only measures blood glucose levels a few times a day. At the same time physiological systems are typically highly nonlinear and stochastic such that recurrent neural networks are suitable models. Typically, such networks are either used purely free running in which the networks predictions are iterated, or in a teacher forcing mode in which actual measurements are substituted

if available. In Section 2 we show that both approaches are problematic for highly stochastic systems and if many realizations of the variable of interest are unknown. The traditional solution is to use a *stochastic* model such as a nonlinear state space model. The problem here is that prediction and training missing data lead to integrals which are usually considered intractable (Lewis, 1986). Alternatively, state dependent linearizations are used for prediction and training, the most popular example being the extended Kalman filter. In this paper we introduce a combination of a nonlinear recurrent neural predictive model and a linear error model which leads to tractable prediction and maximum likelihood adaptation rules. The recurrent neural network can be used in all generality to model the nonlinear dynamics of the system. The only limitation is that the error model is linear which is not a major constraint in many applications. The first advantage of the proposed model is that for single or multiple step *prediction* we obtain simple iteration rules which are a combination of the output of the iterated neural network and a linear Kalman filter which is used for updating the linear error model. The second advantage is that for maximum likelihood *learning* the recurrent neural network can be trained using the real-time recurrent learning rule RTRL and the linear error model can be trained by an EM adaptation rule, implemented using forward-backward Kalman filter equations. We apply our model to develop a model of the glucose/insulin metabolism of a diabetic patient in which blood glucose measurements are only available a few times a day at irregular intervals and compare results from our proposed model to recurrent neural networks trained and used in the free running mode or in the teacher forcing mode as well as to various linear models.

## 2   RECURRENT SYSTEMS WITH MISSING DATA

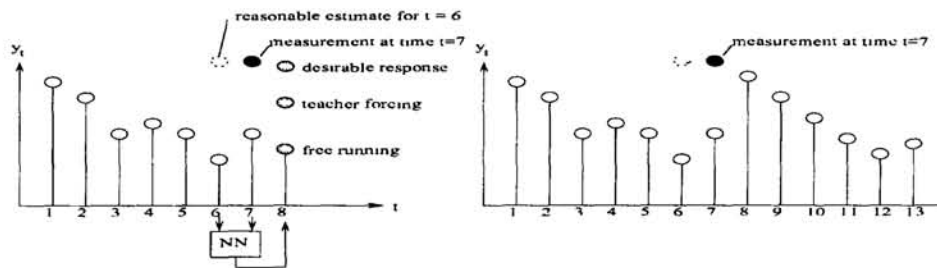

Figure 1: A neural network predicts the next value of a time-series based on the latest two previous measurements (left). As long as no measurements are available ($t = 1$ to $t = 6$), the neural network is iterated (unfilled circles). In a free-running mode, the neural network would ignore the measurement at time $t = 7$ to predict the time-series at time $t = 8$. In a teacher forcing mode, it would substitute the measured value for one of the inputs and use the iterated value for the other (unknown) input. This appears to be suboptimal since our knowledge about the time-series at time $t = 7$ also provides us with information about the time-series at time $t = 6$. For example the dotted circle might be a reasonable estimate. By using the iterated value for the unknown input, the prediction of the teacher forced system is not well defined and will in general lead to unsatisfactory results. A sensible response is shown on the right where the first few predictions after the measurement are close to the measurement. This can be achieved by including a proper error model (see text).

Consider a deterministic nonlinear dynamical model of the form

$$y_t = f_w(y_{t-1}, \ldots, y_{t-N}, u_t)$$

of order $N$, with input $u_t$ and where $f_w(.)$ is a neural network model with parameter-vector $w$. Such a recurrent model is either used in a free running mode in which network predictions are used in the input of the neural network or in a teacher forcing mode where measurements are substituted in the input of the neural network whenever these are available.

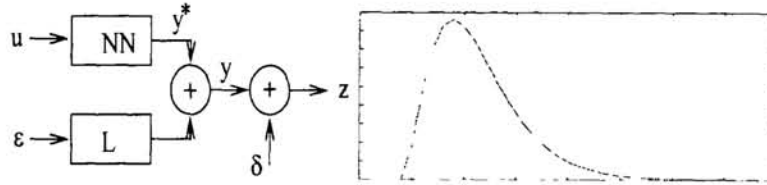

Figure 2: Left: The proposed architecture. Right: Linear impulse response.

Both can lead to undesirable results when many realizations are missing and when the system is highly stochastic. Figure 1 (left) shows that a free running model basically ignores the measurement for prediction and that the teacher forced model substitutes the measured value but leaves the unknown states at their predicted values which also might lead to undesirable responses. The traditional solution is to include a model of the error which leads to nonlinear stochastical models, the simplest being

$$y_t = f_w(y_{t-1}, \ldots, y_{t-N}, u_t) + \epsilon_t$$

where $\epsilon_t$ is assumed to be additive uncorrelated zero-mean noise with probability density $P_\epsilon(\epsilon)$ and represents unmodeled system dynamics. For prediction and learning with missing values we have to integrate over the unknowns which leads to complex integrals which, for nonlinear models, have to be approximated, for example, using Monte Carlo integration.[1] In general, those integrals are computationally too expensive to solve and, in practice, one relies on locally linearized approximations of the nonlinearities typically in form of the extended Kalman filter. The extended Kalman filter is suboptimal and summarizes past data by an estimate of the means and the covariances of the variables involved (Lewis, 1986).

In this paper we pursue an alternative approach. Consider the model with state updates

$$y_t^* = f_w(y_{t-1}^*, \ldots, y_{t-N}^*, u_t) \tag{1}$$

$$x_t = \sum_{i=1}^{K} \theta_i x_{t-i} + \epsilon_t \tag{2}$$

$$y_t = y_t^* + x_t = f_w(y_{t-1}^*, \ldots, y_{t-N}^*, u_t) + \sum_{i=1}^{K} \theta_i x_{t-i} + \epsilon_t \tag{3}$$

and with measurement equation

$$z_t = y_t + \delta_t. \tag{4}$$

where $\epsilon_t$ and $\delta_t$ denote additive noise. The variable of interest $y_t$ is now the sum of the deterministic response of the recurrent neural network $y_t^*$ and a linear system error model $x_t$ (Figure 2). $z_t$ is a noisy measurement of $y_t$. In particular we are interested in the special cases that $y_t$ can be measured with certainty (variance of $\delta_t$ is zero) or that a measurement is missing (variance of $\delta_t$ is infinity). The nice feature is now that $y_t^*$ can be considered a deterministic input to the state space model consisting of the equations (2)– (3). This means that for optimal one-step or multiple-step prediction, we can use the *linear* Kalman filter for equations (2)– (3) and measurement equation (4) by treating $y_t^*$ as deterministic input. Similarly, to train the parameters in the linear part of the system (i.e. $\{\theta_i\}_{i=1}^{N}$) we can use an EM adaptation rule, implemented using forward-backward Kalman filter equations (see the Appendix). The deterministic recurrent neural network is adapted with the residual error which cannot be explained by the linear model, i.e. $target_t^{rnn} = y_t^m - \hat{y}_t^{linear}$

where $y_t^m$ is a measurement of $y_t$ at time $t$ and where $\hat{y}_t^{linear}$ is the estimate of the linear model. After the recurrent neural network is adapted the linear model can be retrained using the residual error which cannot be explained by the neural network, then again the neural network is retrained and so on until no further improvement can be achieved.

The advantage of this approach is that all of the nonlinear interactions are modeled by a recurrent neural network which can be trained deterministically. The linear model is responsible for the noise model which can be trained using powerful learning algorithms for linear systems. The constraint is that the error model cannot be nonlinear which often might not be a major limitation.

## 3   BLOOD GLUCOSE PREDICTION OF A DIABETIC

The goal of this work is to develop a predictive model of the blood glucose of a person with type 1 Diabetes mellitus. Such a model can have several useful applications in therapy: it can be used to warn a person of dangerous metabolic states, it can be used to make recommendations to optimize the person's therapy and, finally, it can be used in the design of a stabilizing control system for blood glucose regulation, a so-called "artificial beta cell" (Tresp, Moody and Delong, 1994). We want the model to be able to adapt using patient data collected under normal every day conditions rather than the controlled conditions typical of a clinic. In a non-clinical setting, only a few blood glucose measurements per day are available.

Our data set consists of the protocol of a diabetic over a period of almost six months. During that time period, times and dosages of insulin injections (basal insulin $u_t^1$ and normal insulin $u_t^2$), the times and amounts of food intake (fast $u_t^3$, intermediate $u_t^4$ and slow $u_t^5$ carbohydrates), the times and durations of exercise (regular $u_t^6$ or intense $u_t^7$) and the blood glucose level $y_t$ (measured a few times a day) were recorded. The $u_t^j, j = 1, \ldots, 7$ are equal to zero except if there is an event, such as food intake, insulin injection or exercise. For our data set, inputs $u_t^j$ were recorded with 15 minute time resolution. We used the first 43 days for training the model (containing 312 measurements of the blood glucose) and the following 21 days for testing (containing 151 measurements of the blood glucose). This means that we have to deal with approximately 93% of missing data during training.

The effects on insulin, food and exercise on the blood glucose are delayed and are approximated by linear response functions. $v_t^j$ describes the effect of input $u_t^j$ on glucose. As an example, the response $v_t^2$ of normal insulin $u_t^2$ after injection is determined by the diffusion of the subcutaneously injected insulin into the blood stream and can be modeled by three first order compartments in series or, as we have done, by a response function of the form $v_t^2 = \sum_\tau g_2(t-\tau)u_\tau^2$ with $g_2(t) = a_2 t^2 e^{-b_2 t}$ (see figure 2 for a typical impulse response). The functional mappings $g_j(.)$ for the digestive tract and for exercise are less well known. In our experiments we followed other authors and used response functions of the above form.

The response functions $g_j(.)$ describe the delayed effect of the inputs on the blood glucose. We assume that the functional form of $g_j(.)$ is sufficient to capture the various delays of the inputs and can be tuned to the physiology of the patient by varying the parameters $a_j, b_j$. To be able to capture the highly nonlinear physiological interactions between the response functions $v_t^j$ and the blood glucose level $y_t$, which is measured only a few times a day, we employ a neural network in combination with a linear error model as described in Section 2. In our experiments $f_w(.)$ is a feedforward multi-layer perceptron with three hidden units. The five inputs to the network were insulin ($in_t^1 = v_t^1 + v_t^2$), food ($in_t^2 = v_t^3 + v_t^4 + v_t^5$), exercise ($in_t^3 = v_t^6 + v_t^7$) and the current and previous estimate of the blood glucose. To be specific, the second order nonlinear neural network model is

$$y_t^* = y_{t-1}^* + f_w\left(y_{t-1}^*, y_{t-2}^*, in_t^1, in_t^2, in_t^3\right) \tag{5}$$

For the linear error model we also use a model of order 2

$$x_t = \theta_1 x_{t-1} + \theta_2 x_{t-2} + \epsilon_t \tag{6}$$

Table 1 shows the explained variance of the test set for different predictive models. [2]

In the first experiment (RNN-FR) we estimate the blood glucose at time $t$ as the output of the neural network $\hat{y}_t = y_t^*$. The neural network is used in the free running mode for training and prediction. We use RTRL to both adapt the weights in the neural network as well as all parameters in the response functions $g_j(.)$. The RNN-FR model explains 14.1 percent of the variance. The RNN-TF model is identical to the previous experiment except that measurements are substituted whenever available. RNN-TF could explain more of the variance (18.8%). The reason for the better performance is, of course, that information about measurements of the blood glucose can be exploited.

The model RNN-LEM2 (error model with order 2) corresponds to the combination of the recurrent neural network and the linear error model as introduced in Section 2. Here, $y_t = x_t + y_t^*$ models the blood glucose and $z_t = y_t + \delta_t$ is the measurement equation where we set the variance of $\delta_t = 0$ for a measurement of the blood glucose at time $t$ and the variance of $\delta_t = \infty$ for missing values. For $\epsilon_t$ we assume Gaussian independent noise. For prediction, equation (5) is iterated in the free running mode. The blood glucose at time $t$ is estimated using a linear Kalman filter, treating $y_t^*$ as deterministic input in the state space model $y_t = x_t + y_t^*$, $z_t = y_t + \delta_t$. We adapt the parameters in the linear error model (i.e. $\theta_1$, $\theta_2$, the variance of $\epsilon_t$) using an EM adaptation rule, implemented using forward-backward Kalman filter equations (see Appendix). The parameters in the neural network are adapted using RTRL exactly the same way as in the RNN-FR model, except that the target is now $target_t^{rnn} = y_t^m - \hat{y}_t^{linear}$ where $y_t^m$ is a measurement of $y_t$ at time $t$ and where $\hat{y}_t^{linear}$ is the estimate of the linear error model (based on the linear Kalman filter). The adaptation of the linear error model and the neural network are performed alternatingly until no significant further improvement in performance can be achieved.

As indicated in Table 1, the RNN-LEM2 model achieves the best prediction performance with an explained variance of 44.9% (first order error model RNN-LEM1: 43.7%). As a comparison, we show the performance of just the linear error model LEM (this model ignores all inputs), a linear model (LM-FR) without an error model trained with RTRL and a linear model with an error model (LM-LEM). Interestingly, the linear error model which does not see any of the inputs can explain more variance (12.9%) than the LM-FR model (8.9%). The LM-LEM model, which can be considered a combination of both can explain more than the sum of the individual explained variances (31.5%) which indicates that the combined training gives better performance than training both submodels individually. Note also, that the nonlinear models (RNN-FR, RNN-TF, RNN-LEM) give considerably better results than their linear counterparts, confirming that the system is highly nonlinear.

Figure 3 (left) shows an example of the responses of some of the models. We see that the free running neural network (dotted line) has relatively small amplitudes and cannot predict the three measurements very well. The RNN-TF model (dashed line) shows a better response to the measurements than the free running network. The best prediction of all measurements is indeed achieved by the RNN-LEM model (continuous line).

Based on the linear iterated Kalman filter we can calculate the variance of the prediction. As shown in Figure 3 (right) the standard deviation is small right after a measurement is available and then converges to a constant value. Based on the prediction and the estimated variance, it will be possible to do a risk analysis for the diabetic (i.e a warning of dangerous metabolic states).

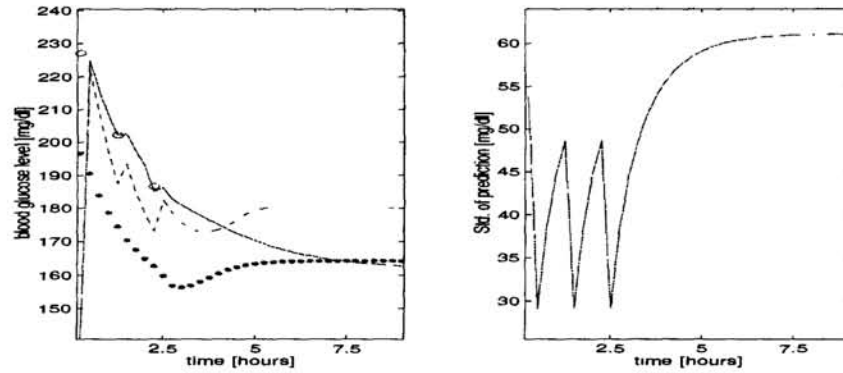

Figure 3: Left: Responses of some models to three measurements. Note, that the prediction of the first measurement is bad for all models but that the RNN-LEM model (continuous line) predicts the following measurements much better than both the RNN-FR (dotted) and the RNN-TF (dashed) model. Right: Standard deviation of prediction error of RNN-LEM.

Table 1: Explained variance on test set [in percent]: $100 \cdot \left(1 - \frac{\text{MSPE}(model)}{\text{MSPE}(mean)}\right)$

| MODEL | % | MODEL | % |
|-------|---|-------|---|
| mean | 0 | RNN-TF | 18.8 |
| LM | 8.9 | LM-LEM | 31.4 |
| LEM | 12.9 | RNN-LEM1 | 43.7 |
| RNN-FR | 14.1 | RNN-LEM2 | 44.9 |

## 4 CONCLUSIONS

We introduced a combination of a nonlinear recurrent neural network and a linear error model. Applied to blood glucose prediction it gave significantly better results than both recurrent neural networks alone and various linear models. Further work might lead to a predictive model which can be used by a diabetic on a daily bases. We believe that our results are very encouraging. We also expect that our specific model can find applications in other stochastical nonlinear systems in which measurements are only available at irregular intervals such that in wastewater treatment, chemical process control and various physiological systems. Further work will include error models for the input measurements (for example, the number of food calories are typically estimated with great uncertainty).

**Appendix: EM Adaptation Rules for Training the Linear Error Model**

Model and observation equations of a general model are[3]

$$x_t = \Theta x_{t-1} + \epsilon_t \quad z_t = M_t x_t + \delta_t. \tag{7}$$

where $\Theta$ is the $K \times K$ transition matrix of the $K$-order linear error model. The $K \times 1$ noise terms $\epsilon_t$ are zero-mean uncorrelated normal vectors with common covariance matrix $Q$. $\delta_t$ is $m$-dimensional [4] zero-mean uncorrelated normal noise vector with covariance matrix $R_t$. Recall that we consider certain measurements and missing values as special cases of

noisy measurements. The initial state of the system is assumed to be a normal vector with mean $\mu$ and covariance $\Sigma$.

We describe the EM equations for maximizing the likelihood of the model. Define the estimated parameters at the $(r+1)$st iterate of EM as the values $\mu, \Sigma, \Theta, Q$ which maximize

$$G(\mu, \Sigma, \Theta, Q) = \mathrm{E}_r(\log \mathrm{L}|z_1, \ldots, z_n) \tag{8}$$

where $\log \mathrm{L}$ is log-likelihood of the complete data $x_0, x_1, \ldots, x_n, z_1, \ldots, z_n$ and $\mathrm{E}_r$ denotes the conditional expectation relative to a density containing the $r$th iterate values $\mu(r), \Sigma(r), \Theta(r)$ and $Q(r)$. Recall that missing targets are modeled implicitly by the definition of $M_t$ and $R_t$.

For calculating the conditional expectation defined in (8) the following set of recursions are used (using standard Kalman filtering results, see (Jazwinski, 1970)). First, we use the forward recursion

$$
\begin{aligned}
x_t^{t-1} &= \Theta x_{t-1}^{t-1} \\
P_t^{t-1} &= \Theta P_{t-1}^{t-1} \Theta^\mathsf{T} + Q \\
K_t &= P_t^{t-1} M_t^\mathsf{T} (M_t P_t^{t-1} M_t^\mathsf{T} + R_t)^{-1} \\
x_t^t &= x_t^{t-1} + K_t(y_t^* - M_t x_t^{t-1}) \\
P_t^t &= P_t^{t-1} - K_t M_t P_t^{t-1}
\end{aligned}
\tag{9}
$$

where we take $x_0^0 = \mu$ and $P_0^0 = \Sigma$. Next, we use the backward recursion

$$
\begin{aligned}
J_{t-1} &= P_{t-1}^{t-1} \Theta^\mathsf{T} (P_t^{t-1})^{-1} \\
x_{t-1}^n &= x_{t-1}^{t-1} + J_{t-1}(x_t^n - \Theta x_{t-1}^{t-1}) \\
P_{t-1}^n &= P_{t-1}^{t-1} + J_{t-1}(P_t^n - P_t^{t-1})J_{t-1}^\mathsf{T} \\
P_{t-1,t-2}^n &= P_{t-1}^{t-1} J_{t-2}^\mathsf{T} + J_{t-1}(P_{t,t-1}^n - \Theta P_{t-1}^{t-1})J_{t-2}^\mathsf{T}
\end{aligned}
\tag{10}
$$

with initialization $P_{n,n-1}^n = (I - K_n M_n)\Theta P_{n-1}^{n-1}$. One forward and one backward recursion completes the E-step of the EM algorithm.

To derive the M-step first realize that the conditional expectations in (8) yield to the following equation:

$$
\begin{aligned}
G = {}& -\tfrac{1}{2}\log|\Sigma| - \tfrac{1}{2}\mathrm{tr}\{\Sigma^{-1}(P_0^n + (x_0^n - \mu)(x_0^n - \mu)^\mathsf{T})\} \\
& -\tfrac{n}{2}\log|Q| - \tfrac{1}{2}\mathrm{tr}\{Q^{-1}(C - B\Theta^\mathsf{T} - \Theta B^\mathsf{T} - \Theta A\Theta^\mathsf{T})\} \\
& -\tfrac{n}{2}\log|R_t| - \tfrac{1}{2}\mathrm{tr}\{R_t^{-1}\sum_{t=1}^n[(y_t^* - M_t x_t)(y_t^* - M_t x_t)^\mathsf{T} + M_t P_t^n M_t^\mathsf{T}]\}
\end{aligned}
\tag{11}
$$

where $\mathrm{tr}\{.\}$ denotes the trace, $A = \sum_{t=1}^n (P_{t-1}^n + x_{t-1}^n x_{t-1}^{n\mathsf{T}})$,

$B = \sum_{t=1}^n (P_{t,t-1}^n + x_t^n x_{t-1}^{n\mathsf{T}})$ and $C = \sum_{t=1}^n (P_t^n + x_t^n x_t^{n\mathsf{T}})$.

$\Theta(r+1) = BA^{-1}$ and $Q(r+1) = n^{-1}(C - BA^{-1}B^\mathsf{T})$ maximize the log-likelihood equation (11). $\mu(r+1)$ is set to $x_0^n$ and $\Sigma$ may be fixed at some reasonable baseline level. The derivation of these equations can be found in (Shumway & Stoffer, 1981).

The E- (forward and backward Kalman filter equations) and M-steps are alternated repeatedly until convergence to obtain the EM solution.

## Footnotes

*{volker.tresp, thomas.briegel}@mchp.siemens.de

[1] For maximum likelihood learning of linear models we obtain EM equations which can be solved using forward-backward Kalman equations (see Appendix).

[2]MSPE(*model*) is the mean squared prediction error on the test set of the model and MSPE(*mean*) is the mean squared prediction error of predicting the mean.

[3]Note, that any linear system of order $K$ can be transformed into a first order linear system of dimension $K$.

[4]$m$ indicates the dimension of the output of the time-series.

## References

Jazwinski, A. H. (1970) *Stochastic Processes and Filtering Theory*, Academic Press, N.Y.

Lewis, F. L. (1986) *Optimal Estimation*, John Wiley, N.Y.

Shumway, R. H. and Stoffer, D. S. (1981) *Time Series Smoothing and Forecasting Using the EM Algorithm*, Technical Report No. 27, Division of Statistics, UC Davis.

Tresp, V., Moody, J. and Delong, W.-R. (1994) *Neural Modeling of Physiological Processes*, in Comput. Learning Theory and Natural Learning Sys. 2, S. Hanson *et al.*, eds., MIT Press.
